# Active Learning
# in the Drug Discovery Process

**Manfred K. Warmuth** [*], **Gunnar Rätsch** [†], **Michael Mathieson** [§],
**Jun Liao** [**], **Christian Lemmen** [‡]

[*] [§] [**] Computer Science Dep., Univ. of Calif. at Santa Cruz
[†] FHG FIRST, Kekuléstr. 7, Berlin, Germany
[‡] DuPont Pharmaceuticals,150 California St. San Francisco.

{manfred,mathiesm,liaojun}@cse.ucsc.edu, Gunnar.Raetsch@anu.edu.au,
clemmen@biosolveit.de

## Abstract

We investigate the following data mining problem from Computational
Chemistry: From a large data set of compounds, find those that bind to
a target molecule in as few iterations of biological testing as possible. In
each iteration a comparatively small batch of compounds is screened for
binding to the target. We apply active learning techniques for selecting
the successive batches.

One selection strategy picks unlabeled examples closest to the maximum
margin hyperplane. Another produces many weight vectors by running
perceptrons over multiple permutations of the data. Each weight vector
votes with its $\pm$ prediction and we pick the unlabeled examples for which
the prediction is most evenly split between $+$ and $-$. For a third selec-
tion strategy note that each unlabeled example bisects the version space
of consistent weight vectors. We estimate the volume on both sides of
the split by bouncing a billiard through the version space and select un-
labeled examples that cause the most even split of the version space.

We demonstrate that on two data sets provided by DuPont Pharmaceu-
ticals that all three selection strategies perform comparably well and are
much better than selecting random batches for testing.

## 1 Introduction

Two of the most important goals in Computational Drug Design are to find *active com-
pounds* in large databases quickly and (usually along the way) to obtain an interpretable
model for what makes a specific subset of compounds active. Activity is typically defined

---

[*]All but last author received partial support from NSF grant CCR 9821087

[†]Current address: Australian National University, Canberra, Austrialia. Partially supported by
DFG (JA 379/9-1, MU 987/1-1) and travel grants from EU (Neurocolt II).

[‡]Current address: BioSolveIT GmbH, An der Ziegelei 75, Sankt Augustin, Germany

as binding to a target molecule. Most of the time an iterative approach to the problem is employed. That is in each iteration a batch of unlabeled compounds is screened against the target using some sort of biological assay[MGST97]. The desired goal is that many active *hits* show up in the assays of the selected batches.

From the Machine Learning point of view all examples (compounds) are initially unlabeled. In each iteration the learner selects a batch of un-labeled examples for being labeled as positive (active) or negative (inactive). In Machine Learning this type of problem has been called "query learning" [Ang88] "selective sampling" [CAL90] or "active learning" [TK00]. A *Round0* data set contains 1,316 chemically diverse examples, only 39 of which are positive. A second *Round1* data set has 634 examples with 150 positives.[1] This data set is preselected on the basis of medicinal chemistry intuition. Note that our classification problem is fundamentally asymmetric in that the data sets have typically many more negative examples and the Chemists are more interested in the positive hits because these compounds might lead to new drugs. What makes this problem challenging is that each compound is described by a vector of 139,351 binary shape features. The vectors are sparse (on the average 1378 features are set per Round0 compound and 7613 per Round1 compound).

We are working with retrospective data sets for which we know all the labels. However, we simulate the real-life situation by initially hiding all labels and only giving to the algorithm the labels for the requested batches of examples (virtual screening). The long-term goal of this type of research is to provide a computer program to the Chemists which will do the following interactive job: At any point new unlabeled examples may be added. Whenever a test is completed, the labels are given to the program. Whenever a new test needs to be set up, the Chemist asks the program to suggest a batch of unlabeled compounds. The suggested batch might be "edited" and augmented using the invaluable knowledge and intuition of the medicinal Chemist. The hope is that the computer assisted approach allows for mining larger data sets more quickly. Note that compounds are often generated with virtual Combinatorial Chemistry. Even though compound descriptors can be computed, the compounds have not been synthesized yet. In other words it is comparatively easy to generate lots of unlabeled data.

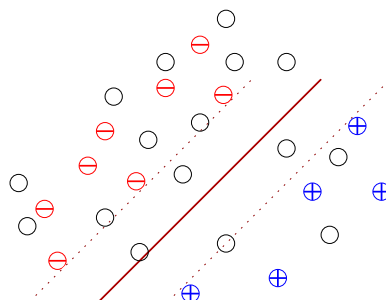

**Figure 1:** Three types of compounds/points: ⊕ are active, ⊖ are inactive and ◯ are yet unlabeled. The *Maximum Margin Hyperplane* is used as the internal classifier.

In our case the Round0 data set consists of compounds from Vendor catalog and corporate collections. Much more design effort went into the harder Round1 data set. Our initial results are very encouraging. Our selection strategies do much better than choosing random batches indicating that the long-term goal outlined above may be feasible.

Thus from the Machine Learning point of view we have a fixed set of points in $\mathcal{R}^{139,351}$ that are either unlabeled or labeled positive or negative. (See Figure 1). The binary descriptors of the compounds are rather "complete" and the data is always linearly separable. Thus we concentrate on simple linear classifiers in this paper.[2] We analyzed a large number of different ways to produce hyperplanes and combine hyperplanes. In the next section we describe different selection strategies on the basis of these hyperplanes in detail and provide an experimental comparison. Finally in Section 3 we give some theoretical justification for why the strategies are so effective.

## 2  Different Selection Criteria and their Performance

A selection algorithm is specified in three parts: a batch size, an initialization and a selection strategy. In practice it is not cost effective to test single examples at a time. We always chose 5% of the total data set as our batch size, which matches reasonably with typical experimental constraints. The initial batches are chosen at random until at least one positive and one negative example are found. Typically this is achieved with the first batch. All further batches are chosen using the selection strategy.

As we mentioned in the introduction, all our selection strategies are based on linear classifiers of the data labeled so far. All examples are normalized to unit-length and we consider homogeneous hyperplanes $\{\mathbf{x} : \mathbf{w} \cdot \mathbf{x} = 0\}$ where the normal direction $\mathbf{w}$ is again unit-length. A plane $\mathbf{w}$ predicts with $\text{sign}(\mathbf{w} \cdot \mathbf{x})$ on the example/compound $\mathbf{x}$.

Once we specify how the weight vector is found then the next batch is found by selecting the unlabeled examples *closest* to this hyperplane. The simplest way to obtain such a weight vector is to run a perceptron over the labeled data until it produces a consistent weight vector (Perc). Our second selection strategy (called SVM) uses the maximum margin hyperplane [BGV92] produced by a Support Vector Machine. When using the perceptron to predict for example handwritten characters, it has been shown that "voting" the $\pm$ predictions of many hyperplanes improves the predictive performance [FS98]. So we always start from the weight vector zero and do multiple passes over the data until the perceptron is consistent. After processing each example we store the weight vector. We remember all weight vectors for each pass[3] and do this for 100 random permutations of the labeled examples. Each weight vector gets one $\pm$ vote. The prediction on an example is positive if the total vote is larger than zero and we select the unlabeled examples whose total vote is *closest* to zero[4]. We call this selection strategy VoPerc.

The dot product is commutative. So when $\mathbf{w} \cdot \mathbf{x} > 0$ then the point $\mathbf{x}$ lies on the positive side of the hyperplane $\mathbf{w}$. In a *dual* view the *point* $\mathbf{w}$ lies on the positive side of the *hyperplane* $\mathbf{x}$ (Recall all instances and weight vectors have unit-length). A weight vector $\mathbf{w}$ that is consistent with all $\pm$-labeled examples $(\mathbf{x}_n, y_n)$ must lie on the $y_n$-side of the plane $\mathbf{x}_n$ for all $n$. The set of all consistent weight vectors is called the *version space* which is a section of the unit hypersphere bounded by the planes corresponding to the labeled examples. An unlabeled hyperplane $\mathbf{x}_n$ bisects the version space. For our third selection strategy (VolEst) a billiard is bounced 1000 times inside the version space and the fraction $f_n$ of bounce points on the positive side of $\mathbf{x}_n$ is computed. The prediction for $\mathbf{x}_n$ is positive if $f_n$ is larger than half and the strategy selects unlabeled points whose fraction is *closest* to half.

In Figure 2 (left) we plot the true positives and false positives w.r.t. the whole data set for Perc and VoPerc showing that VoPerc performs slightly better. Also VoPerc has lower variance (Figure 2 (right)). Figure 3 (left) shows the averaged true positives and false positives of VoPerc, SVM, and VolEst. We note that all three perform similarly. We also plotted ROC curves after each batch has been added (not shown). These plots also show that all three strategies are comparable.

The three strategies VoPerc, SVM, and VolEst all perform much better than the corresponding strategies where the selection criterion is to select random unlabeled examples instead of using a "closest" criterion. For example we show in Figure 4 that SVM is significantly better than SVM-Rand. Surprisingly the improvement is larger on the easier Round0 data set. The reason is that the Round0 has a smaller fraction of positive examples (3%). Recall

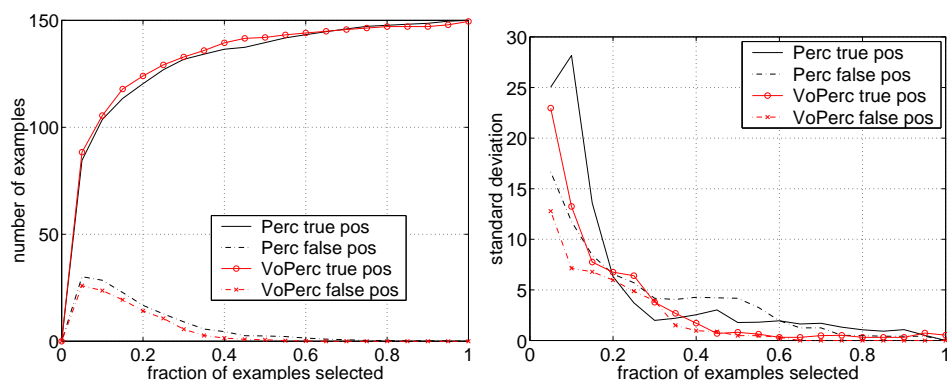

Figure 2: (left) Average (over 10 runs) of true positives and false positives on the entire Round1 data set after each 5% batch for Perc and VoPerc. (right) Standard deviation over 10 runs.

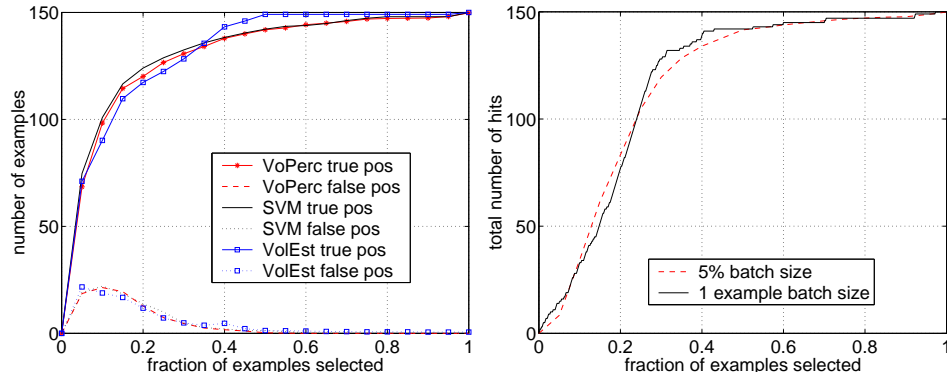

Figure 3: (left) Average (over 10 runs) of true and false positives on entire Round1 data set after each 5% batch for VoPerc, SVM, and VolEst. (right) Comparison of 5% batch size and 1 example batch size for VoPerc on Round1 data.

that the Round1 data was preselected by the Chemists for actives and the fraction was raised to about 25%. This suggest that our methods are particularly suitable when few positive examples are hidden in a large set of negative examples.

The simple strategy SVM of choosing unlabeled examples closest to the maximum margin hyperplane has been investigated by other authors (in [CCS00] for character recognition and in [TK00] for text categorization). The labeled points that are closest to the hyperplane are called the support vectors because if all other points are removed then the maximum margin hyperplane remains unchanged. In Figure 5 we visualize the location of the points in relation to the center of the hyperplane. We show the location of the points projected onto the normal direction of the hyperplane. For each 5% batch the location of the points is scattered onto a thin stripe. The hyperplane crosses the stripe in the middle. In the left plot the distances are scaled so that the support vectors are at distance $\pm 1$. In the right plot the geometric distance to the hyperplane is plotted. Recall that we pick unlabeled points closest to the hyperplane (center of the stripe). As soon as the "window" between the support vectors is cleaned most positive examples have been found (compare with the SVM curves given in Figure 3 (left)). Also shrinking the width of the geometric window corresponds to improved generalization.

So far our three selection strategies VoPerc, SVM and VolEst have shown similar performance. The question is whether the performance criterion considered so far is suitable for the drug design application. Here the goal is to label/verify many positive compounds quickly. We therefore think that the total number of positives (hits) among all examples tested so far is the best performance criterion. Note that the total number of hits of the random selection strategy grows linearly with the number of batches (In each random batch

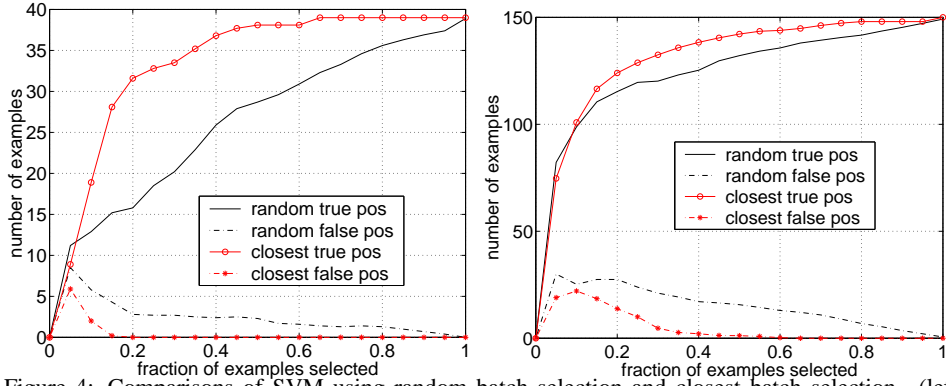

Figure 4: Comparisons of SVM using random batch selection and closest batch selection. (left) Round0 data. (right) Round1 data.

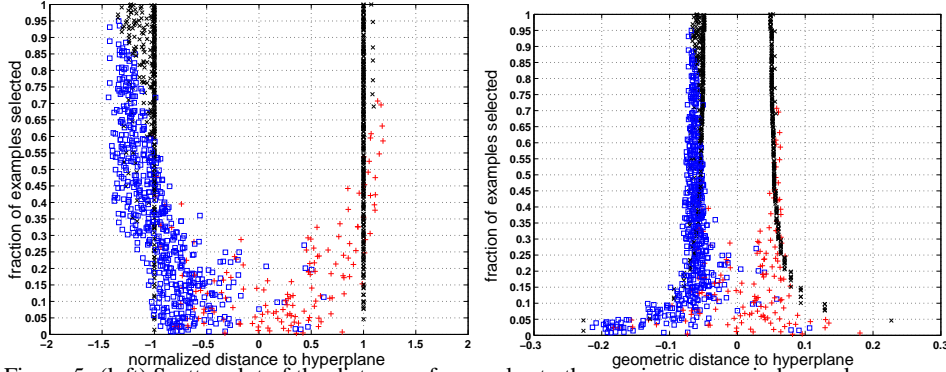

Figure 5: (left) Scatter plot of the distance of examples to the maximum margin hyperplane normalized so support vectors are at $\pm 1$. (right) Scatter plot of the geometric distance of examples to the hyperplane. Each stripe shows location of a random sub-sample of points (Round1 data) after an additional 5% batch has been labeled by SVM. Selected examples are black x, unselected positives are red plus, unselected negatives are blue square.

we expect 5% hits). In contrast the total number of hits of VoPerc, SVM and VolEst is 5% in the first batch (since it is random) but much faster thereafter (See Figure 6). VoPerc performs the best.

Since the positive examples are much more valuable in our application, we also changed the selection strategy SVM to selecting unlabeled examples of largest *positive distance*[5] $\mathbf{w} \cdot \mathbf{x}$ to the maximum margin hyperplane $\mathbf{w}$ (SVM$^+$) rather than smallest distance $|\mathbf{w} \cdot \mathbf{x}|$. Correspondingly VoPerc$^+$ picks the unlabeled example with the highest vote and VolEst$^+$ picks the unlabeled example with the largest fraction $f_n$. The total hit plots of the resulting modified strategies SVM$^+$, VoPerc$^+$ and VolEst$^+$ are improved ( see Figure 7 versus Figure 6 ). However the generalization plots of the modified strategies (i.e. curves like Figure 3(left)) are slightly worse for the new versions. Thus in some sense the original strategies are better at "exploration" (giving better generalization on the entire data set) while the modified strategies are better at "exploitation" (higher number of total hits). We show this trade-off in Figure 8 for SVM and SVM$^+$. The same trade-off occurs for the VoPerc($^+$) and VolEst($^+$) pairs of strategies(not shown).

Finally we investigate the effect of batch size on performance. For simplicity we only show total hit plots for VoPerc( Figure 3 (right) ). Note that for our data a batch size of 5% (31 examples for Round1) is performing not much worse than the experimentally unrealistic batch size of only 1 example. Only when the results for batch size 1 are much better than

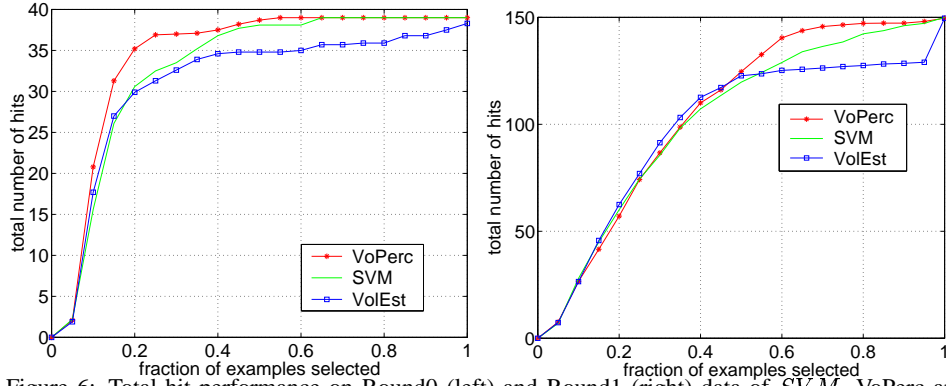

Figure 6: Total hit performance on Round0 (left) and Round1 (right) data of $SVM$, VoPerc and VolEst with 5% batch size.

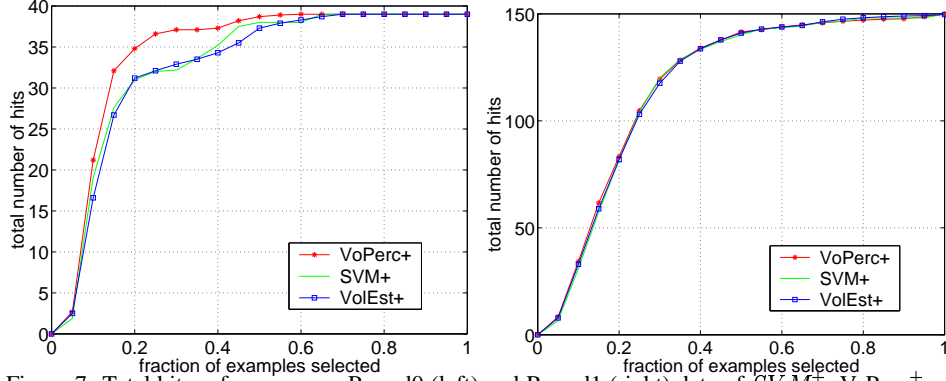

Figure 7: Total hit performance on Round0 (left) and Round1 (right) data of $SVM^+$, VoPerc$^+$ and VolEst$^+$ with 5% batch size.

the results for larger batch sizes, more sophisticated selection strategies are worth exploring that pick say a batch that is "close" and at the same time "diverse".

At this point our data sets are still small enough that we were able to precompute all dot products (the kernel matrix). After this preprocessing, one pass of a perceptron is at most $O(NM)$, where $N$ is the number of labeled examples and $M$ the number of mistakes. Finding the maximum margin hyperplane is estimated at $O(N^{2.5})$ time. For the computation of VolEst we need to spend $O(N^2)$ per bounce of the billiard. In our implementations we used SVM Light [Joa99] and the billiard algorithm of [Ruj97, RM00, HGC99].

If we have the internal hypothesis of the algorithm then for applying the selection criterion we need to evaluate the hypothesis for each unlabeled point. This cost is proportional to the number of support vectors for the SVM-based methods and proportional to the number of mistakes for the perceptron-based methods. In the case of VolEst we again need $O(N^2)$ time per bounce, where $N$ is the number of labeled points.

Overall VolEst was clearly the slowest. For much larger data sets VoPerc seems to be the simplest and the most adaptable.

## 3 Theoretical Justifications

As we see in Figure 5(right) the geometric margin of the support vectors (half the width of the window) is shrinking as more examples are labeled. Thus the following goal is reasonable for designing selection strategies: pick unlabeled examples that cause the margin to shrink the most. The simplest such strategy is to pick examples closest to the maximum margin hyperplane since these example are expected to change the maximum margin

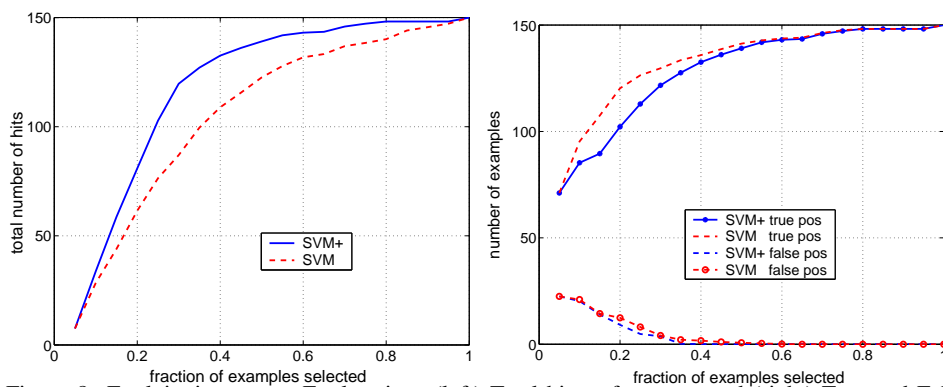

Figure 8: Exploitation versus Exploration: (left) Total hit performance and (right) True and False positives performance (right) of SVM and $SVM^+$ on Round 1 data

hyperplane the most [TK00, CCS00].

An alternative goal is to reduce the volume of the version space. This volume is a rough measure of the remaining uncertainty in the data. Recall that both the weight vectors and instances have unit length. Thus $\mathbf{w} \cdot \mathbf{x}$ is the distance of the point $\mathbf{x}$ to the plane $\mathbf{w}$ as well as (in the dual view) the distance of the point $\mathbf{w}$ to the plane $\mathbf{x}$. The maximum margin hyperplane $\mathbf{w}$ is the point $\mathbf{w}$ in version space with the largest sphere that is completely contained in the version space [Ruj97, RM00]. After labeling $\mathbf{x}$ only one side of the plane $\mathbf{x}$ remains. So if $\mathbf{x}$ passes close to the point $\mathbf{w}$ then about half of the largest sphere is eliminated from the version space. So this is a second justification for selecting unlabeled examples closest to the maximum margin hyperplane.

Our selection strategy VolEst starts from any point inside the version space and then bounces a billiard 1000 times. The billiard is almost always ergodic (See discussion in [Ruj97]). Thus the fraction $f_n$ of bounces on the positive side of an unlabeled hyperplane $\mathbf{x}_n$ is an estimate of the fraction of volume on the positive side of $\mathbf{x}_n$. Since it is unknown how $\mathbf{x}_n$ will be labeled, the best example are those that split the version space in half. Thus in VolEst we select unlabeled points for which $f_n$ is closest to half. The thinking underlying our strategy VolEst is most closely related to the Committee Machine where $k$ random concepts in the version space are asked to vote on the next random example and the label of that example is requested only if the vote is close to an even split [SOS92].

We tried to improve our estimate of the volume by replacing $f_n$ by the fraction of the total trajectory located on the positive side of $\mathbf{x}_n$. On our two data sets this did not improve the performance (not shown). We also averaged the 1000 bounce points. The resulting weight vector $\mathbf{w}$ (an approximation to the center of mass of the version space) approximates the so called *Bayes point* [Ruj97] which has the following property: Any unlabeled hyperplane passing through the Bayes point $\mathbf{w}$ cuts the version space roughly [6] in half. We thus tested a selection strategy which picks unlabeled points closest to the estimated center of mass. This strategy was again indistinguishable from the other two strategies based on bouncing the billiard.

We have no rigorous justification for the $^+$ variants of our algorithms.

## 4    Conclusion

We showed how the active learning paradigm ideally fits the drug design cycle. After some deliberations we concluded that the total number of positive examples (hits) among the tested examples is the best performance criterion for the drug design application. We found

that a number of different selection strategies with comparable performance. The variants that select the unlabeled examples with the highest score (i.e. the $^+$ variants) perform better. Overall the selection strategies based on the Voted Perceptron were the most versatile and showed slightly better performance.

## Footnotes

[1]Data provided by DuPont Pharmaceuticals.

[2]On the current data sets kernels did not improve the results (not shown).

[3]Surprisingly with some smart bookkeeping this can all be done with essentially no computational overhead. [FS98]

[4]Instead of voting the predictions of all weight vectors one can also average all the weight vectors after normalizing them and select unlabeled examples closest to the resulting single weight vector. This way of averaging leads to slightly worse results (not shown).

[5]In Figure 5 this means we are selecting from right to left

[6]Even in dimension two there is no point that does this exactly [Ruj97].

# References

[Ang88] D. Angluin. Queries and concept learning. *Machine Learning*, 2:319–342, 1988.

[BGV92] B.E. Boser, I.M. Guyon, and V.N. Vapnik. A training algorithm for optimal margin classifiers. In D. Haussler, editor, *Proceedings of the 5th Annual ACM Workshop on Computational Learning Theory*, pages 144–152, 1992.

[CAL90] D. Cohn, L. Atlas, and R. Ladner. Training connectionist networks with queries and selective sampling. *Advances in Neural Information Processing Systems*, 2:566–573, 1990.

[CCS00] C. Campbell, N. Cristianini, and A. Smola. Query learning with large margin classifiers. In *Proceedings of ICML2000*, page 8, Stanford, CA, 2000.

[FS98] Y. Freund and R. Schapire. Large margin classification using the perceptron algorithm. In *Proc. 11th Annu. Conf. on Comput. Learning Theory*. ACM Press, New York, NY, July 1998.

[HGC99] Ralf Herbrich, Thore Graepel, and Colin Campbell. Bayes point machines: Estimating the bayes point in kernel space. In *Proceedings of IJCAI Workshop Support Vector Machines*, pages 23–27, 1999.

[Joa99] T. Joachims. Making large–scale SVM learning practical. In B. Schölkopf, C.J.C. Burges, and A.J. Smola, editors, *Advances in Kernel Methods — Support Vector Learning*, pages 169–184, Cambridge, MA, 1999. MIT Press.

[MGST97] P. Myers, J. Greene, J. Saunders, and S. Teig. Rapid, reliable drug discovery. *Today's Chemist at Work*, 6:46–53, 1997.

[RM00] P. Ruján and M. Marchand. Computing the bayes kernel classifier. In *Advances in Large Margin Classifiers*, volume 12, pages 329–348. MIT Press, 2000.

[Ruj97] P. Ruján. Playing billiard in version space. *Neural Computation*, 9:99–122, 1997.

[SOS92] H. Seung, M. Opper, and H. Sompolinsky. Query by committee. In *Proceedings of the Fifth Workshop on Computational Learning Theory*, pages 287–294, 1992.

[TK00] S. Tong and D. Koller. Support vector machine active learning with applications to text classification. In *Proceedings of the Seventeenth International Conference on Machine Learning*, San Francisco, CA, 2000. Morgan Kaufmann.
